# The Storage Capacity
# of a Fully-Connected Committee Machine

**Yuansheng Xiong**
Department of Physics, Pohang Institute of Science and Technology,
Hyoja San 31, Pohang, Kyongbuk, Korea
xiong@galaxy.postech.ac.kr

**Chulan Kwon**
Department of Physics, Myong Ji University,
Yongin, Kyonggi, Korea
ckwon@wh.myongji.ac.kr

**Jong-Hoon Oh**
Lucent Technologies, Bell Laboratories,
600 Mountain Ave., Murray Hill, NJ07974, U. S. A.
jhoh@physics.bell-labs.com

## Abstract

We study the storage capacity of a fully-connected committee machine with a large number $K$ of hidden nodes. The storage capacity is obtained by analyzing the geometrical structure of the weight space related to the internal representation. By examining the asymptotic behavior of order parameters in the limit of large $K$, the storage capacity $\alpha_c$ is found to be proportional to $K\sqrt{\ln K}$ up to the leading order. This result satisfies the mathematical bound given by Mitchison and Durbin, whereas the replica-symmetric solution in a conventional Gardner's approach violates this bound.

## 1 INTRODUCTION

Since Gardner's pioneering work on the storage capacity of a single layer perceptron[1], there have been numerous efforts to use the statistical mechanics formulation to study feed-forward neural networks. The storage capacity of multi-layer neural networks has been of particular interest, together with the generalization problem. Barkai, Hansel and Kanter[2] studied a parity machine with a

non-overlapping receptive field of continuous weights within a one-step replica symmetry breaking (RSB) scheme, and their result agrees with a mathematical bound previously found by Mitchison and Durbin (MD)[3]. Subsequently Barkai, Hansel and Sompolinsky[4] and Engel et al.[5] have studied the committee machine, which is closer to the multi-layer perceptron architecture and is most frequently used in real-world applications. Though they have derived many interesting results, particularly for the case of a finite number of hidden units, it was found that their the replica-symmetric (RS) result violates the MD bound in the limit where the number of hidden units $K$ is large.

Recently, Monasson and O'Kane[6] proposed a new statistical mechanics formalism which can analyze the weight-space structure related to the internal representations of hidden units. It was applied to single layer perceptrons[7, 8, 9] as well as multi-layer networks[10, 11, 12]. Monasson and Zecchina[10] have successfully applied this formalism to the case of both committee and parity machines with non-overlapping receptive fields (NRF)[10]. They suggested that analysis of the RS solution under this new statistical mechanics formalism can yield results just as good as the one-step RSB solution in the conventional Gardner's method.

In this letter, we apply this formalism for a derivation of the storage capacity of a fully-connected committee machine, which is also called a committee machine with overlapping receptive field (ORF) and is believed to be a more relevant architecture. In particular, we obtain the value of the critical storage capacity in the limit of large $K$, which satisfies the MD bound. It also agrees with a recent one-step RSB calculation, using the conventional Gardner method, to within a small difference of a numerical prefactor[13]. Finally we will briefly discuss the fully-connected parity machine.

## 2  WEIGHT SPACE STRUCTURE OF THE COMMITTEE MACHINE

We consider a fully-connected committee machine with $N$ input units, $K$ hidden units and one output unit, where weights between the hidden units and the output unit are set to 1. The network maps input vectors $\{x_i^\mu\}$, where $\mu = 1, ..., P$, to output $y^\mu$ as:

$$
\begin{aligned}
y^\mu &= \text{sgn}\left(\sum_{j=1}^{K} h_j^\mu\right) \\
&= \text{sgn}\left[\sum_{j=1}^{K} \text{sgn}\left(\sum_{i=1}^{N} W_{ji} x_i^\mu\right)\right],
\end{aligned}
\tag{1}
$$

where $W_{ji}$ is the weight between the $i$th input node and the $j$th hidden unit. $h_j^\mu \equiv \text{sgn}(\sum_{i=1}^{N} W_{ji} x_i^\mu)$ is the $j$th component of the internal representation for input pattern $\{x_i^\mu\}$. We consider continuous weights with spherical constraint, $\sum_i^N W_{ji} = N$.

Given $P = \alpha N$ patterns, the learning process in a layered neural network can be interpreted as the selection of cells in the weight space corresponding to a set of suitable internal representations $\mathbf{h} = \{h_j^\mu\}$, each of which has a non-zero elementary

volume defined by:

$$V_{\mathbf{h}} = \mathrm{Tr}_{\{W_{j,}\}} \prod_{\mu} \Theta \left( y^{\mu} \sum_{j} h_j^{\mu} \right) \prod_{\mu,j} \Theta \left( h_j^{\mu} \sum_{i} W_{ji} x_i^{\mu} \right), \qquad (2)$$

where $\Theta(x)$ is the Heaviside step function. The Gardner's volume $V_G$, that is, the volume of the weight space which satisfies the given input-output relations, can be written as the sum of the cells over all internal representations:

$$V_G = \sum_{\mathbf{h}} V_{\mathbf{h}}. \qquad (3)$$

The method developed by Monasson and his collaborators [6, 10] is based on analysis of the detailed internal structure, that is, how the Gardner's volume $V_G$ is decomposed into elementary volumes $V_{\mathbf{h}}$ associated with a possible internal representation. The distribution of the elementary volumes can be derived from the free energy,

$$g(r) = -\frac{1}{Nr} \left\langle\!\!\left\langle \ln \left( \sum_{\mathbf{h}} V_{\mathbf{h}}^r \right) \right\rangle\!\!\right\rangle, \qquad (4)$$

where $\langle\!\langle \cdots \rangle\!\rangle$ denotes the average over patterns. The entropy $\mathcal{N}[w(r)]$ of the volumes whose average sizes are equal to $w(r) = -1/N \ln \langle\!\langle V_{\mathbf{h}} \rangle\!\rangle$, can be given by the Legendre relations

$$\mathcal{N}[w(r)] = -\frac{\partial g(r)}{\partial (1/r)}, \qquad w(r) = \frac{\partial [rg(r)]}{\partial r} \qquad (5)$$

respectively.

The entropies $\mathcal{N}_D = \mathcal{N}[w(r=1)]$ and $\mathcal{N}_R = \mathcal{N}[w(r=0)]$ are of most importance, and will be discussed below. In the thermodynamic limit, $1/N \langle\!\langle \ln(V_G) \rangle\!\rangle = -g(r=1)$ is dominated by elementary volumes of size $w(r=1)$, of which there are $\exp(N\mathcal{N}_D)$. Furthermore, the most numerous elementary volumes have the size $w(r=0)$ and number $\exp(N\mathcal{N}_R)$. The vanishing condition for the entropies is related to the zero volume condition for $V_G$ and thus gives the storage capacity. We focus on the entropy $\mathcal{N}_D$ of elementary volumes dominating the weight space $V_G$.

## 3   ORDER PARAMETERS AND PHASE TRANSITION

For a fully-connected machine, the overlaps between different hidden units should be taken into account, which makes this problem much more difficult than the tree-like (NRF) architecture studied in Ref. [10]. The replicated partition function for the fully-connected committee machine reads:

$$\left\langle\!\!\left\langle \left( \sum_{\mathbf{h}} V_{\mathbf{h}}^r \right)^n \right\rangle\!\!\right\rangle = \left\langle\!\!\left\langle \mathrm{Tr}_{h_j^{\mu\alpha}} \mathrm{Tr}_{W_j^{\mu\alpha}} \prod_{\mu\alpha} \Theta \left( \sum_{j} h_j^{\mu\alpha} \right) \prod_{\mu j \alpha a} \Theta \left( h_j^{\mu\alpha} \sum_{i} W_{ji}^{\alpha a} x_i^{\mu} \right) \right\rangle\!\!\right\rangle, \qquad (6)$$

with $a = 1, \cdots, r$ and $\alpha = 1, \cdots, n$. Unlike Gardner's conventional approach, we need two sets of replica indices for the weights. We introduce the order parameters,

$$Q_{jk}^{\alpha\beta ab} = \frac{1}{N} \sum_{i} W_{ji}^{\alpha a} W_{ki}^{\beta b}, \qquad (7)$$

where the indices $a, b$ originate from the integer power $r$ of elementary volumes, and $\alpha, \beta$ are the standard replica indices. The replica symmetry ansatz leads to five

order parameters as:

$$Q_{jk}^{\alpha\beta ab} = \begin{cases} q^* & (j=k, \alpha=\beta, a \neq b), \\ q & (j=k, \alpha \neq \beta), \\ c & (j \neq k, \alpha=\beta, a=b), \\ d^* & (j \neq k, \alpha=\beta, a \neq b), \\ d & (j \neq k, \alpha \neq \beta), \end{cases} \tag{8}$$

where $q^*$ and $q$ are, respectively, the overlaps between the weight vectors connected to the same hidden unit of the same ($\alpha = \beta$) and different ($\alpha \neq \beta$) replicas corresponding to the two different internal representations. The order parameters $c, d^*$ and $d$ describe the overlaps between weights that are connected to different hidden units, of which $c$ and $d^*$ are the overlaps within the same replica whereas $d$ correlates different replicas.

Using a standard replica trick, we obtain the explicit form of $g(r)$. One may notice that the free energy evaluated at $r=1$ is reduced to the RS results obtained by the conventional method on the committee machine[4, 5], which is independent of $q^*$ and $d^*$. This means that the internal structure of the weight space is overlooked by conventional calculation of the Gardner's volume. When we take the limit $r \to 1$, the free energy can be expanded as:

$$g(r, q^*, q, c, d^*, d) = g(1, q, c, d) + (r-1)\frac{\partial g(r, q^*, q, c, d^*, d)}{\partial r}\bigg|_{r=1}. \tag{9}$$

As noticed, $g(r, q^*, q, c, d^*, d)$ is the same as the RS free energy in the Gardner's method. From the relation:

$$\mathcal{N}_D = -\frac{\partial g(r)}{\partial(1/r)}\bigg|_{r=1} = \frac{\partial g(r)}{\partial r}\bigg|_{r=1}, \tag{10}$$

we obtain the explicit form of $\mathcal{N}_D$.

In the case of the NRF committee machine, where each of the hidden units is connected to different input units, we do not have a phase transition. Instead, a single solution is applicable for the whole range of $\alpha$. In contrast, the phase-space structure of the fully-connected committee machine is more complicated than that of the NRF committee machine. When a small number of input patterns are given, the system is in the permutation-symmetry (PS) phase[4, 5, 14], where the role of each hidden unit is not specialized. In the PS phase, the Gardner's volume is a single connected region. The order parameters associated with different hidden units are equal to the corresponding ones associated with the same hidden unit. When a critical number of patterns is given, the Gardner's volume is divided into many islands, each one of which can be transformed into other ones by permutation of hidden units. This phenomenon is called permutation symmetry breaking (PSB), and is usually accompanied by a first-order phase transition. In the PSB phase, the role of each hidden unit is specialized to store a larger number of patterns effectively. A similar breaking of symmetry has been observed in the study of generalization[14, 15], where the first-order phase transition induces discontinuity of the learning curve. It was pointed out that the critical storage capacity is attained in the PSB phase[4, 5], and our recent one-step replica symmetry breaking calculation confirmed this picture[13]. Therefore, we will focus on the analysis of the PSB solution near the storage capacity, in which $q^*, q \to 1$, and $c, d^*, d$ are of order $1/K$.

## 4  STORAGE CAPACITY

When we analyze the results for free energy, the case with $q(r = 1)$, $c(r = 1)$ and $d(r = 1)$ is reduced to the usual saddle-point solutions of the replica symmetric expression of the Gardner's volume $g(r = 1)$[4, 5]. When $K$ is large, the trace over all allowed internal representations can be evaluated similarly to Ref.[4]. The saddle-point equations for $q^*$ and $d^*$ are derived from the derivative of the free energy in the limit $r \to 1$, as in Eq. (9). The details of the self-consistent equations are not shown for space consideration. In the following, we only summarize the asymptotic behavior of the order parameters for large $\alpha$:

$$1 - q + d - c \sim \frac{128}{(\pi - 2)^2} \frac{K^2}{\alpha^2}, \tag{11}$$

$$1 - q + (K - 1)(c - d) \sim \frac{32}{\pi - 2} \frac{K}{\alpha^2}, \tag{12}$$

$$q + (K - 1)d \sim \frac{\pi - 2}{\alpha}, \tag{13}$$

$$1 - q^* + d^* - c \sim \frac{\pi^2 \Gamma^2}{2\alpha^2}, \tag{14}$$

$$1 - q^* + (K - 1)(c - d^*) \sim \frac{\pi^2 \Gamma^2}{2\alpha^2}, \tag{15}$$

where $\Gamma = -[\sqrt{\pi} \int du\, H(u) \ln H(u)]^{-1} \simeq 0.62$.

It is found that all the overlaps between weights connecting different hidden units have scaling of $-1/K$, whereas the typical overlaps between weights connecting the same hidden unit approach one. The order parameters $c$, $d$ and $d^*$ are negative, showing antiferromagnetic correlations between different hidden units, which implies that each hidden unit attempts to store patterns different from those of the others[4, 5].

Finally, the asymptotic behavior of the entropy $\mathcal{N}_D$ in the large $K$ limit can be derived using the scaling given above. Near the storage capacity, $\mathcal{N}_D$ can be written, up to the leading order, as:

$$\mathcal{N}_D \simeq K \ln K - \frac{(\pi - 2)^2 \alpha^2}{256 K}. \tag{16}$$

Being the entropy of a discrete system, $\mathcal{N}_D$ cannot be negative. Therefore, $\mathcal{N}_D = 0$ gives an indication of the upper bound of storage capacity, that is, $\alpha_c \sim \frac{16}{\pi - 2} K \sqrt{\ln K}$. The storage capacity per synapse, $\frac{16}{\pi - 2} \sqrt{\ln K}$, satisfies the rigorous bound $\sim \ln K$ derived by Mitchison and Durbin (MD)[3], whereas the conventional RS result[4, 5], which scales as $\sqrt{K}$, violates the MD bound.

## 5  DISCUSSIONS

Recently, we have studied this problem using a conventional Gardner approach in the one-step RSB scheme[13]. The result yields the same scaling with respect to $K$, but a coefficient smaller by a factor $\sqrt{2}$. In the present paper, we are dealing with the fine structure of version space related to internal representations. On the other hand, the RSB calculation seems to handle this fine structure in association with symmetry breaking between replicas. Although the physics of the two approaches seems to be somehow related, it is not clear which of the two can yield a better

estimate of the storage capacity. It is possible that the present RS calculation does not properly handle the RSB picture of the system. Monasson and his co-workers reported that the Almeida-Thouless instability of the RS solutions decreases with increasing $K$, in the NRF case[10, 11]. A similar analysis for the fully-connected case certainly deserves further research. On the other hand, the one-step RSB scheme also introduces approximation, and possibly it cannot fully explain the weight-space structure associated with internal representations.

It is interesting to compare our result with that of the NRF committee machine along the same lines[10]. Based on the conventional RS calculation, Angel et al. suggested that the same storage capacity per synapse for both fully-connected and NRF committee machines will be similar, as the overlap between the hidden nodes approaches zero.[5]. While the asymptotic scaling with respect to $K$ is the same, the storage capacity in the fully-connected committee machine is larger than in the NRF one. It is also consistent with our result from one-step RSB calculation[13]. This implies that the small, but nonzero negative correlation between the weights associated with different hidden units, enhances the storage capacity. This may be good news for those people using a fully connected multi-layer perceptron in applications.

From the fact that the storage capacity of the NRF parity machine is $\ln K / \ln 2$[2, 10], which saturates the MD bound, one may guess that the storage capacity of a fully-connected parity machine is also proportional to $K \ln K$. It will be interesting to check whether the storage capacity per synapse of the fully-connected parity machine is also enhanced compared to the NRF machine[16].

## Acknowledgements

This work was partially supported by the Basic Science Special Program of POSTECH and the Korea Ministry of Education through the POSTECH Basic Science Research Institute(Grant No. BSRI-96-2438). It was also supported by non-directed fund from Korea Research Foundation, 1995, and by KOSEF grant 971-0202-010-2.

# References

[1] E. Gardner, Europhys, Lett. 4(4), 481 (1987); E. Gardner, J. Phys. A21, 257 (1988); E. Gardner and B. Derrida, J. Phys. A21, 271 (1988).

[2] E. Barkai, D. Hansel and I. Kanter, Phys. Rev. Lett. V 65, N18, 2312 (1990).

[3] G. J. Mitchison and R. M. Durbin, Boil. Cybern. 60, 345 (1989).

[4] E. Barkai, D. Hansel and H. Sompolinsky, Phys. Rev. E45, 4146 (1992).

[5] A. Engel, H. M. Köhler, F. Tschepke, H. Vollmayr, and A. Zippeelius, Phys. Rev. E45, 7590 (1992).

[6] R. Monasson and D. O'Kane, Europhys. Lett. 27, 85(1994).

[7] B. Derrida, R. B. Griffiths and A Prugel-Bennett, J. Phys. A 24, 4907 (1991).

[8] M. Biehl and M. Opper, *Neural Networks: The Statistical Mechanics Perspective*, Jong-Hoon Oh, Chulan Kwon, and Sungzoon Cho (eds.) (World Scientific, Singapore, 1995).

[9] A. Engel and M. Weigt, Phys. Rev. E53, R2064 (1996).

[10] R. Monasson and R. Zecchina, Phys. Rev. Lett. 75, 2432 (1995); 76, 2205 (1996).

[11] R. Monasson and R. Zecchina, Mod. Phys. B, Vol.9, 1887-1897 (1996).

[12] S. Cocco, R. Monasson and R. Zecchina, Phys. Rev. E54, 717 (1996).

[13] C. Kwon and J. H. Oh, J. Phys. A, in press.

[14] K. Kang, J. H. Oh, C. Kwon and Y. Park, Phys. Rev. E48, 4805 (1993).

[15] H. Schwarze and J. Hertz, Europhys. Lett. 21, 785 (1993).

[16] Y. Xiong, C. Kwon and J.-H. Oh, to be published (1997).
